# Limitations of self-organizing maps for vector quantization and multidimensional scaling

**Arthur Flexer**

The Austrian Research Institute for Artificial Intelligence
Schottengasse 3, A-1010 Vienna, Austria
and
Department of Psychology, University of Vienna
Liebiggasse 5, A-1010 Vienna, Austria
arthur@ai.univie.ac.at

## Abstract

The limitations of using self-organizing maps (SOM) for either clustering/vector quantization (VQ) or multidimensional scaling (MDS) are being discussed by reviewing recent empirical findings and the relevant theory. SOM's remaining ability of doing both VQ *and* MDS at the same time is challenged by a new combined technique of online $K$-means clustering plus Sammon mapping of the cluster centroids. SOM are shown to perform significantly worse in terms of quantization error, in recovering the structure of the clusters and in preserving the topology in a comprehensive empirical study using a series of multivariate normal clustering problems.

## 1 Introduction

Self-organizing maps (SOM) introduced by [Kohonen 84] are a very popular tool used for visualization of high dimensional data spaces. SOM can be said to do clustering/vector quantization (VQ) *and* at the same time to preserve the spatial ordering of the input data reflected by an ordering of the code book vectors (cluster centroids) in a one or two dimensional output space, where the latter property is closely related to multidimensional scaling (MDS) in statistics. Although the level of activity and research around the SOM algorithm is quite large (a recent overview by [Kohonen 95] contains more than 1000 citations), only little comparison among the numerous existing variants of the basic approach and also to more traditional statistical techniques of the larger frameworks of VQ and MDS is available. Additionally, there is only little advice in the literature about how to properly use

SOM in order to get optimal results in terms of either vector quantization (VQ) or multidimensional scaling or maybe even both of them. To make the notion of SOM being a tool for "data visualization" more precise, the following question has to be answered: Should SOM be used for doing VQ, MDS, both at the same time or none of them?

Two recent comprehensive studies comparing SOM either to traditional VQ *or* MDS techniques separately seem to indicate that SOM is not competitive when used for either VQ or MDS: [Balakrishnan et al. 94] compare SOM to $K$-means clustering on 108 multivariate normal clustering problems with known clustering solutions and show that SOM performs significantly worse in terms of data points misclassified[1], especially with higher numbers of clusters in the data sets. [Bezdek & Nikhil 95] compare SOM to principal component analysis and the MDS-technique Sammon mapping on seven artificial data sets with different numbers of points and dimensionality and different shapes of input distributions. The degree of preservation of the spatial ordering of the input data is measured via a Spearman rank correlation between the distances of points in the input space and the distances of their projections in the two dimensional output space. The traditional MDS-techniques preserve the distances much more effectively than SOM, the performance of which decreases rapidly with increasing dimensionality of the input data.

Despite these strong empirical findings that speak against the use of SOM for either VQ or MDS there remains the appealing ability of SOM to do both VQ *and* MDS at the same time. It is the aim of this work to find out, whether a combined technique of traditional vector quantization (clustering) *plus* MDS on the code book vectors (cluster centroids) can perform better than Kohonen's SOM on a series of multivariate normal clustering problems in terms of quantization error (mean squared error), recovering the cluster structure (Rand index) and preserving the topology (Pearson correlation). All the experiments were done in a rigoruos statistical design using multiple analysis of variance for evaluation of the results.

## 2    SOM and vector quantization/clustering

A vector quantizer (VQ) is a mapping, $q$, that assigns to each input vector $x$ a reproduction (code book) vector $\hat{x} = q(x)$ drawn from a finite reproduction alphabet $\hat{A} = \{\hat{x}_i, i = 1, \dots, N\}$. The quantizer $q$ is completely described by the reproduction alphabet (or codebook) $\hat{A}$ together with the partition $S = \{S_i, i = 1, \dots, N\}$, of the input vector space into the sets $S_i = \{x : q(x) = \hat{x}_i\}$ of input vectors mapping into the $i^{th}$ reproduction vector (or code word) [Linde et al. 80]. To be compareable to SOM, our VQ assigns to each of the input vectors $x = (x^0, x^1, \dots, x^{k-1})$ a socalled code book vector $\hat{x} = (\hat{x}^0, \hat{x}^1, \dots, \hat{x}^{k-1})$ of the same dimensionality $k$. For reasons of data compression, the number of code book vectors $N \ll n$, where $n$ is the number of input vectors.

Demanded is a VQ that produces a mapping $q$ for which the expected distortion caused by reproducing the input vectors $x$ by code book vectors $q(x)$ is at least locally minimal. The expected distortion is usually esimated by using the average distortion $D$, where the most common distortion measure is the squared-error

distortion $d$:

$$D = \frac{1}{n} \sum_{i=0}^{n-1} d(x_i, q(x_i)) \quad (1) \qquad d(x, \hat{x}) = \sum_{i=0}^{k-1} \mid x_i - \hat{x}_i \mid^2 \quad (2)$$

The classical vector quantization technique to achieve such a mapping is the LBG-algorithm [Linde et al. 80], where a given quantizer is iteratively improved. Already [Linde et al. 80] noted that their proposed algorithm is almost similar to the $k$-means approach developed in the cluster analysis literature starting from [MacQueen 67]. Closely related to SOM is online $K$-means clustering (oKMC) consisting of the following steps:

1. Initialization: Given $N$ = number of code book vectors, $k$ = dimensionality of the vectors, $n$ = number of input vectors, a training sequence $\{x_j; j = 0, \dots, n-1\}$, an initial set $\hat{A}_0$ of $N$ code book vectors $\hat{x}$ and a discrete-time coordinate $t = 0 \dots, n-1$.

2. Given $\hat{A}_t = \{\hat{x}_i; i = 1, \dots, N\}$, find the minimum distortion partition $P(\hat{A}_t) = \{S_i; i = 1, \dots, N\}$. Compute $d(x_t, \hat{x}_i)$ for $i = 1, \dots, N$. If $d(x_t, \hat{x}_i) \leq (x_t, \hat{x}_l)$ for all $l$, then $x_t \in S_i$.

3. Update the code book vector with the minimum distortion

$$\hat{x}_{(t)}(S_i) = \hat{x}_{(t-1)}(S_i) + \alpha[x_{(t)} - \hat{x}_{(t-1)}(S_i)] \quad (3)$$

   where $\alpha$ is a learning parameter to be defined by the user. Define $\hat{A}_{t+1} = \hat{x}(P(\hat{A}_t))$, replace $t$ by $t + 1$, if $t = n - 1$, halt. Else go to step 2.

The main difference between the SOM-algorithm and oKMC is the fact that the code book vectors are ordered either on a line or on a planar grid (i.e. in a one or two dimensional output space). The iterative procedure is the same as with oKMC where formula (3) is replaced by

$$\hat{x}_{(t)}(S_i) = \hat{x}_{(t-1)}(S_i) + h[x_{(t)} - \hat{x}_{(t-1)}(S_i)] \quad (4)$$

and this update is not only computed for the $\hat{x}_i$ that gives minimum distortion, but also for all the code book vectors which are in the neighbourhood of this $\hat{x}_i$ on the line or planar grid. The degree of neighbourhood and amount of code book vectors which are updated together with the $\hat{x}_i$ that gives minimum distortion is expressed by $h$, a function that decreases both with distance on the line or planar grid and with time and that also includes an additional learning parameter $\alpha$. If the degree of neighbourhood is decreased to zero, the SOM-algorithm becomes equal to the oKMC-algorithm.

Whereas local convergence is guaranteed for oKMC (at least for decreasing $\alpha$, [Bottou & Bengio 95]), no general proof for the convergence of SOM with nonzero neighbourhood is known. [Kohonen 95, p.128] notes that the last steps of the SOM algorithm should be computed with zero neighbourhood in order to guarantee "the most accurate density approximation of the input samples".

## 3  SOM and multidimensional scaling

Formally, a topology preserving algorithm is a transformation $\Phi : R^k \mapsto R^p$, that either preserves *similarities* or just *similarity orderings* of the points in the input space $R^k$ when they are mapped into the outputspace $R^p$. For most algorithms it is the case that both the number of input vectors $\mid x \in R^k \mid$ and the number of output

vectors $| \hat{x} \in R^p |$ are equal to $n$. A transformation $\Phi : \hat{x} = \Phi(x)$, that preserves *similarities* poses the strongest possible constraint since $d(x_i, x_j) = \hat{d}(\hat{x}_i, \hat{x}_j)$ for all $x_i, x_j \in R^k$, all $\hat{x}_i, \hat{x}_j \in R^p$, $i, j = 1, \ldots, n-1$ and $d$ $(\hat{d})$ being a measure of distance in $R^k$ $(R^p)$. Such a transformation is said to produce an *isometric* image.

Techniques for finding such transformations $\Phi$ are, among others, various forms of *multidimensional scaling*[2] (MDS) like metric MDS [Torgerson 52], nonmetric MDS [Shepard 62] or Sammon mapping [Sammon 69], but also principal component analysis (PCA) (see e.g. [Jolliffe 86]) or SOM. Sammon mapping is doing MDS by minimizing the following via steepest descent:

$$\frac{1}{\sum_{i=0}^{n-1} \sum_{j<i} d(x_i, x_j)} \sum_{i=0}^{n-1} \sum_{j<i} \frac{(d(x_i, x_j) - \hat{d}(\hat{x}_i, \hat{x}_j))^2}{d(x_i, x_j)} \quad (5)$$

Since the SOM has been designed heuristically and not to find an extremum for a certain cost or energy function[3] and the theoretical connection to the other MDS algorithms remains unclear. It should be noted that for SOM the number of output vectors $| \hat{x} \in R^p |$ is limited to $N$, the number of cluster centroids $\hat{x}$ and that the $\hat{x}$ are further restricted to lie on a planar grid. This restriction entails a discretization of the outputspace $R^p$.

# 4   Online $K$-means clustering plus Sammon mapping of the cluster centroids

Our new combined approach consists of simply finding the set of $\hat{A} = \{\hat{x}_i, i = 1, \ldots, N\}$ code book vectors that give the minimum distortion partition $P(\hat{A}) = \{S_i; i = 1, \ldots, N\}$ via the oKMC algorithm and then using the $\hat{x}_i$ as input vectors to Sammon mapping and thereby obtaining a two dimensional representation of the $\hat{x}_i$ via minimizing formula (5). Contrary to SOM, this two dimensional representation is not restricted to any fixed form and the distances between the $N$ mapped $\hat{x}_i$ directly correspond to those in the original higher dimension. This combined algorithm is abbreviated oKMC+.

# 5   Empirical comparison

The empirical comparison was done using a 3 factorial experimental design with 3 dependent variables. The multivariate normal distributions were generated using the procedure by [Milligan & Cooper 85], which since has been used for several comparisons of cluster algorithms (see e.g. [Balakrishnan et al. 94]). The marginal normal distributions gave internal cohesion of the clusters by warranting that more than 99% of the data lie within 3 standard deviations ($\sigma$). External isolation was defined as having the first dimension nonoverlapping by truncating the normal distributions in the first dimension to $\pm 2\sigma$ and defining the cluster centroids to be $4.5\sigma$ apart. In all other dimensions the clusters were allowed to overlap by setting the distance per dimension between two centroids randomly to lie between $\pm 6\sigma$. The data was normalized to zero mean and unit variance in all dimensions.

| algorithm | no. clusters | dimension | msqe | Rand | corr. |
|---|---|---|---|---|---|
| SOM | 4 | 4 | 0.53 | 1.00 | 0.64 |
| | | 6 | 1.53 | 0.91 | 0.72 |
| | | 8 | 1.15 | 0.99 | 0.74 |
| | 9 | 4 | 0.33 | 0.97 | 0.48 |
| | | 6 | 0.54 | 0.97 | 0.66 |
| | | 8 | 0.81 | 0.96 | 0.74 |
| | mean SOM | | **0.81** | **0.97** | **0.67** |
| oKMC+ | 4 | 4 | 0.53 | 0.99 | 0.87 |
| | | 6 | 1.06 | 0.99 | 0.87 |
| | | 8 | 1.17 | 1.00 | 0.91 |
| | 9 | 4 | 0.29 | 0.98 | 0.89 |
| | | 6 | 0.47 | 0.99 | 0.87 |
| | | 8 | 0.56 | 0.98 | 0.86 |
| | mean oKMC+ | | **0.68** | **0.99** | **0.88** |

*Factor 1, Type of algorithm:* The number of code book vectors of both the SOM and the oKMC+ were set equal to the number of clusters known to be in the data. The SOMs were planar grids consisting of $2 \times 2$ $(3 \times 3)$ code book vectors. During the first phase (1000 code book updates) $\alpha$ was set to 0.05 and the radius of the neighbourhood to 2 (5). During the second phase (10000 code book updates) $\alpha$ was set to 0.02 and the radius of the neighbourhood to 0 to guarantee the most accurate vector quantization [Kohonen 95, p.128]. The oKMC+ algorithm had the parameter $\alpha$ fixed to 0.02 and was trained using each data set 20 times, the minimization of formula (5) was stopped after 100 iterations. Both SOM and oKMC+ were run 10 times on each data set and only the best solutions, in terms of mean squared error, were used for further analysis.

*Factor 2, Number of clusters* was set to 4 and 9.

*Factor 3, Number of dimensions* was set to $4, 6, or 8$.

*Dependent variable 1: mean squared error* was computed using formula (1).

*Dependent variable 2, Rand index* (see [Hubert & Arabie 85]) is a measure of agreement between the true, known partition structure and the obtained clusters. Both the numerator and the denominator of the index reflect frequency counts. The numerator is the number of times a pair of data is either in the same or in different clusters in both known and obtained clusterings for all possible comparisons of data points. Since the denominator is the total number of all possible pairwise comparisons, an index value of 1.0 indicates an exact match of the clusterings.

*Dependent variable 3, correlation* is a measure of the topology preserving abilities of the algorithms. The Pearson correlation of the distances $d(x_1, x_2)$ in the input space and the distances $\hat{d}(\hat{x}_i, \hat{x}_j)$ in the output space for all possible pairwise comparisons of data points is computed. Note that for SOM the coordinates of the code book vectors on the planar grid were used to compute the $\hat{d}$. An algorithm that preserves all distances in every neighbourhood would produce an *isometric* image and yield a value of 1.0 (see [Bezdek & Nikhil 95] for a discussion of measures of topolgy preservation).

For each cell in the full-factorial $2 \times 2 \times 3$ design 3 data sets with 25 points for each cluster were generated resulting in a total of 36 data sets. A multiple analysis of variance (MANOVA) yielded the following significant effects at the .05 error level:

The mean squared error is lower for oKMC+ than for SOM, it is lower for the 9-cluster problem than for the 4-cluster problem and is higher for higher dimensional

data. There is also a combined effect of the number of clusters and dimensions on the mean squared error. The Rand index is higher for oKMC+ than for SOM, there is also a combined effect of the number of clusters and dimensions. The correlation index is higher for oKMC+ than for SOM. Since the main interest of this study is the effect of the type of algorithm on the dependent variables, the mean performances for SOM and oKMC+ are printed in bold letters in the table. Note that the overall differences in the performances of the two algorithms are blurred by the significant effects of the other factors and that therefore the differences of the grand means across the type of algorithms appear rather small. Only by applying a MANOVA, effects of the factor 'type of algorithms' that are masked by additional effects of the other two factors 'number of clusters' and 'number of dimensions' could still be detected.

## 6   Discussion and Conclusion

From the theoretical comparison of SOM to oKMC it should be clear that in terms of quantization error, SOM should only be possible to perform as good as oKMC if SOM's neighbourhood is set to zero. Additional experiments, not reported here in detail for brevity, with nonzero neighbourhood till the end of SOM training gave even worse results since the neighbourhood tends to pull the obtained cluster centroids away from the true ones. The Rand index is only slightly better for oKMC+. The high values indicate that both algorithms were able to recover the known cluster structure. Topology preserving is where SOM performs worst compared to oKMC+. This is a direct implication of the restriction to planar grids which allows only $\sum_{i=2}^{s} i, (s \geq 2)$ different distances in an $s \times s$ planar grid instead of $\frac{N(N-1)}{2}$ different distances for $N = s \times s$ cluster centroids mapped via Sammon mapping in the case of oKMC+. Using a nonzero neighbourhood at the end of SOM training did not warrant any significant improvements.

An argument that could be brought forward against our approach towards comparing SOM and oKMC+ is that it would be unfair or not correct to set the number of SOM's code book vectors equal to the number of clusters known to be in the data. In fact it seems to be common practice to apply SOM with numbers of code book vectors that are a multiple of the input vectors available for training (see e.g. [Kohonen 95, pp.113]). Two things have to be said against such an argumentation: First if one uses more or even only the same amount of code book vectors than input vectors during vector quantization, each code book vector will become identical to one of the input vectors in the limit of learning. So every $x_i$ is replaced with an identical $\hat{x}_i$, which does not make any sense and runs counter to every notion of vector quantization. This means that SOMs employing numbers of code book vectors that are a multiple of the input vectors available can be used for MDS only. But even such big SOMs do MDS in a very crude way: We computed SOMs consisting of either $20 \times 20$ (for data sets consisting of 4 clusters and 100 points) or $30 \times 30$ (for data sets consisting of 9 clusters and 225 points) code book vectors for all 36 data sets which gave an average correlation of 0.77 between the distances $d_i$ and $\hat{d}_i$. This is significantly worse at the .05 error level compared to the average correlation of 0.95 achieved by Sammon mapping applied to the input data directly.

Our data sets consisted of iid multivariate normal distributions which therefore have spherical shape. All VQ algorithms using squared distances as a distortion measure, including our versions of oKMC as well as SOM, are inherently designed for such distributions. Therefore, the clustering problems in this study, being also perfectly seperable in one dimension, were very simple and should be solveable with little or no error by any clustering or MDS algorithm.

In this work we examined the vague concept of using SOM as a "data visualization tool" both from a theoretical and empirical point of view. SOM cannot outperform traditional VQ techniques in terms of quantization error and should therefore not be used for doing VQ. From [Bezdek & Nikhil 95] as well as from our discussion of SOM's restriction to planar grids in the output space which allows only a restricted number of different distances to be represented, it should be evident that SOM is also a rather crude way of doing MDS. Our own empirical results show that if one wants to have an algorithm that does both VQ and MDS at the same time, there exists a very simple combination of traditional techniques (our oKMC+) with wellknown and established properties that clearly outperforms SOM.

Whether it is a good idea to combine clustering or vector quantization and multidimensional scaling at all and whether more principled approaches (see e.g. [Bishop et al. this volume], also for pointers to further related work) can yield even better results than our oKMC+ and last but not least what self-organizing maps *should* be used for under this new light remain questions to be answered by future investigations.

**Acknowledgements:** Thanks are due to James Pardey, University of Oxford, for the Sammon code. The SOM_PAK, Helsinki University of Technology, was used for all computations of self-organizing maps. This work has been started within the framework of the BIOMED-1 concerted action ANNDEE, sponsored by the European Commission, DG XII, and the Austrian Federal Ministry of Science, Transport, and the Arts, which is also supporting the Austrian Research Institute for Artificial Intelligence. The author is supported by a doctoral grant of the Austrian Academy of Sciences.

## Footnotes

[1]Although SOM is an unsupervised technique not built for classification, the number of points missclassified to a wrong cluster center *is* an appropriate and commonly used performance measure for cluster procedures if the true cluster structure is known.

[2]Note that for MDS not the actual coordinates of the points in the input space but only their distances or the ordering of the latter are needed.

[3][Erwin et al. 92] even showed that such an objective function cannot exist for SOM.

# References

[Balakrishnan et al. 94] Balakrishnan P.V., Cooper M.C., Jacob V.S., Lewis P.A.: A study of the classification capabilities of neural networks using unsupervised learning: a comparison with k-means clustering, Psychometrika, Vol. 59, No. 4, 509-525, 1994.

[Bezdek & Nikhil 95] Bezdek J.C., Nikhil R.P.: An index of topological preservation for feature extraction, Pattern Recognition, Vol. 28, No. 3, pp.381-391, 1995.

[Bishop et al. this volume] Bishop C.M., Svensen M., Williams C.K.I.: GTM: A Principled Alternative to the Self-Organizing Map, this volume.

[Bottou & Bengio 95] Bottou L., Bengio Y.: Convergence Properties of the K-Means Algorithms, in Tesauro G., et al.(eds.), Advances in Neural Information Processing System 7, MIT Press, Cambridge, MA, pp.585-592, 1995.

[Erwin et al. 92] Erwin E., Obermayer K., Schulten K.: Self-organizing maps: ordering, convergence properties and energy functions, Biological Cybernetics, 67, 47- 55, 1992.

[Hubert & Arabie 85] Hubert L.J., Arabie P.: Comparing partitions, J. of Classification, 2, 63-76, 1985.

[Jolliffe 86] Jolliffe I.T.: Principal Component Analysis, Springer, 1986.

[Kohonen 84] Kohonen T.: Self-Organization and Associative Memory, Springer, 1984.

[Kohonen 95] Kohonen T.: Self-organizing maps, Springer, Berlin, 1995.

[Linde et al. 80] Linde Y., Buzo A., Gray R.M.: An Algorithm for Vector Quantizer Design, IEEE Transactions on Communications, Vol. COM-28, No. 1, January, 1980.

[MacQueen 67] MacQueen J.: Some Methods for Classification and Analysis of Multivariate Observations, Proc. of the Fifth Berkeley Symposium on Math., Stat. and Prob., Vol. 1, pp. 281-296, 1967.

[Milligan & Cooper 85] Milligan G.W., Cooper M.C.: An examination of procedures for determining the number of clusters in a data set, Psychometrika 50(2), 159-179, 1985.

[Sammon 69] Sammon J.W.: A Nonlinear Mapping for Data Structure Analysis, IEEE Transactions on Comp., Vol. C-18, No. 5, p.401-409, 1969.

[Shepard 62] Shepard R.N.: The analysis of proximities: multidimensional scaling with an unknown distance function. I., Psychometrika, Vol. 27, No. 2, p.125-140, 1962.

[Torgerson 52] Torgerson W.S.: Multidimensional Scaling, I: theory and method, Psychometrika, 17, 401-419, 1952.
